# A Neuromorphic Multi-chip Model of a Disparity Selective Complex Cell

**Eric K. C. Tsang and Bertram E. Shi**
Dept. of Electrical and Electronic Engineering
Hong Kong University of Science and Technology
Kowloon, HONG KONG SAR
{eeeric,eebert}@ust.hk

## Abstract

The relative depth of objects causes small shifts in the left and right retinal positions of these objects, called binocular disparity. Here, we describe a neuromorphic implementation of a disparity selective complex cell using the binocular energy model, which has been proposed to model the response of disparity selective cells in the visual cortex. Our system consists of two silicon chips containing spiking neurons with monocular Gabor-type spatial receptive fields (RF) and circuits that combine the spike outputs to compute a disparity selective complex cell response. The disparity selectivity of the cell can be adjusted by both position and phase shifts between the monocular RF profiles, which are both used in biology. Our neuromorphic system performs better with phase encoding, because the relative responses of neurons tuned to different disparities by phase shifts are better matched than the responses of neurons tuned by position shifts.

## 1  Introduction

The accurate perception of the relative depth of objects enables both biological organisms and artificial autonomous systems to interact successfully with their environment. Binocular disparity, the positional shift between corresponding points in two eyes or cameras caused by the difference in their vantage points, is one important cue that can be used to infer depth.

In the mammalian visual system, neurons in the visual cortex combine signals from the left and right eyes to generate responses selective for a particular disparity [1]. Ohzawa et al.[2] proposed the binocular energy model to explain the responses of binocular complex cells in the cat visual cortex, and found that the predictions of this model are in good agreement with measured data. This model also matches data from the macaque [3].

In the energy model, a neuron achieves its particular disparity tuning by either a position or a phase shift between its monocular receptive field (RF) profiles for the left and right eyes. Based on an analysis of a population of binocular cells, Anzai et al. [4] suggest that the cat primarily encodes disparity via a phase shift, although position shifts may play a larger role at higher spatial frequencies. Computational studies show that it is possible to estimate disparity from the relative responses of model complex cells tuned to different disparities [5][6].

This paper describes a neuromorphic implementation of disparity tuned neurons constructed according to the binocular energy model. Section 2 reviews the binocular energy model and the encoding of disparity by position and phase shifts. Section 3 describes our implementation. Section 4 presents measured results from the system illustrating better performance for neurons tuned by phase than by position. This preference arises because the position-tuned neurons are more sensitive to the mismatch in the circuits on the Gabor-type filter chip than the phase-tuned neurons. We have characterized the mismatch on the chip, as well as its effect on the complex cell outputs, and found that the phase model least sensitive to the parameters that vary most. Section 5 summarizes our results.

## 2  The Binocular Energy Model

Ohzawa et al. [2] proposed the binocular energy model to explain the response of binocular complex cells measured in the cat. Anzai et al. further refined the model in a series of papers [4][7][8]. In this model, the response of a binocular complex cell is the linear combination of the outputs of four binocular simple cells, as shown in Figure 1. The response of a binocular simple cell is computed by applying a linear binocular filter to the input from the two eyes, followed by a half squaring nonlinearity: $r_s = (|b(x_R, x_L, \phi_R, \phi_L)|^+)^2$ where $|b|^+ = \max\{b, 0\}$ is the positive half-wave rectifying nonlinearity. The linear binocular filter output is the sum of two monocular filter outputs

$$b(c_R, c_L, \phi_R, \phi_L) = m(c_R, \phi_R, I_R) + m(c_L, \phi_L, I_L) \tag{1}$$

where the monocular filters linearly combine image intensity, $I(x)$, with a Gabor receptive field profile

$$m(c, \phi, I) = \Sigma_x g(x, c, \phi) I(x)$$

$$g(x, c, \phi) = \kappa e^{-\frac{1}{2}(x-c)^T C^{-1}(x-c)} \cos(\Omega^T(x-c) + \phi)$$

where $x \in \mathbb{Z}^2$ indexes pixel position. The subscripts R and L denote parameters or image intensities from the right or left eye. The parameters $\Omega \in \mathbb{R}^2$ and $C \in \mathbb{R}^{2 \times 2}$ control the spatial frequency and bandwidth of the filter and $\kappa$ controls the gain. These parameters are assumed to be the same in all of the simple cells that make up a complex cell. However, the center position, $c \in \mathbb{R}^2$ and the phase $\phi \in \mathbb{R}$ vary, both between the two eyes and among the four simple cells.

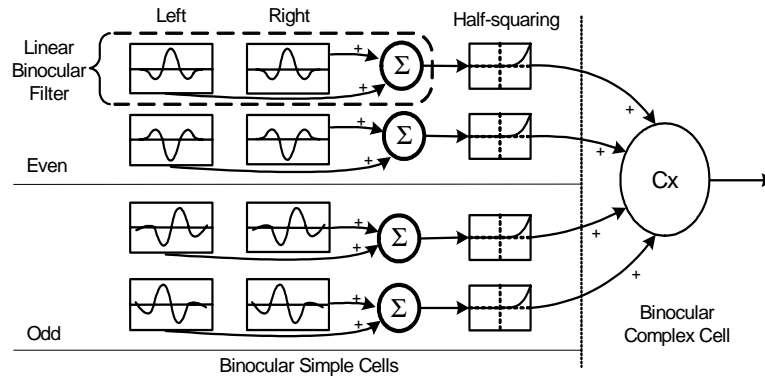

Fig. 1: Binocular energy model of a complex cell.

While the response of simple cells depends heavily upon the stimulus phase and contrast, the response of complex cells is largely independent of the phase and contrast. The binocular energy model posits that complex cells achieve this invariance by linearly combining the outputs of four simple cell responses whose binocular filters are in quadrature phase, being identical except that they differ in phase by $\pi/2$. Because filters that differ in phase by $\pi$ are identical except for a change in sign, we only require two unique binocular filters, the four required simple cell outputs being obtained by positive and negative half squaring their outputs.

Complex cells constructed according to the binocular energy model respond to disparities in the direction orthogonal to their preferred orientation. Their disparity tuning in this direction depends upon the relative center positions and the relative phases of the monocular filters. A binocular complex cell whose monocular filters are shifted by $\Delta c = c_R - c_L$ and $\Delta\phi = \phi_R - \phi_L$ will respond maximally for an input disparity $D_{pref} \approx \Delta c - \Delta\phi/\Omega$ (i.e. $I_R(x) \approx I_L(x - D_{pref})$). Disparity is encoded by a position shift if $\Delta c \neq 0$ and $\Delta\phi = 0$. Disparity is encoded by a phase shift if $\Delta c = 0$ and $\Delta\phi \neq 0$. The cell uses a hybrid encoding if both $\Delta c \neq 0$ and $\Delta\phi \neq 0$. Phase encoding and position encoding are equivalent for the zero disparity tuned cell ($\Delta c = 0$ and $\Delta\phi = 0$).

## 3  Neuromorphic Implementation

Figure 2 shows a block diagram of our binocular cell system, which uses a combination of analog and digital processing. At this time, we use a pattern generator to supply left and right eye input. This gives us precise electronic control over spatial shift between the left and right eye inputs to the orientation selective neurons. We plan to replace the pattern generator with silicon retinae in the future. The left and right eye inputs are processed by two Gabor-type chips that contain retinotopic arrays of spiking neuron circuits whose spatial RF profiles are even and odd symmetric Gabor-type functions. The address filters extract spikes from four neurons in each chip whose output spike rates represent the positive and negative components of the odd and even symmetric filters centered at a desired retinal location. These spike trains are combined in the binocular combination block to implement the summation in (1). The complex cell computation block performs the half squaring nonlinearity and linear summation. In the following, we detail the design of the major building blocks.

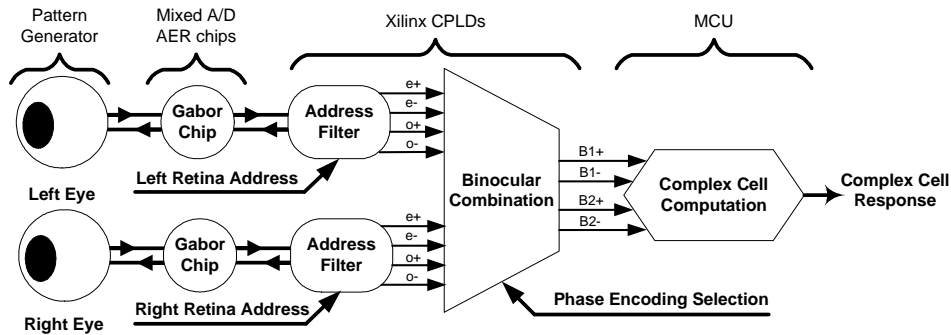

Fig. 2: System block diagram of a neuromorphic complex cell. The opposite direction arrows represent the AER handshaking protocol. The three groups of four parallel arrows represent spiking channels. The labels "e/o" and "+/-" represent EVEN/ODD and ON/OFF. The top labels indicate the type of hardware used to implement each stage.

## 3.1 Gabor-type filtering chip

Images from each eye are passed to a Gabor-type filtering chip [9] that implements the monocular filtering required by the simple cells. Given a spike rate encoded 32 x 64 pixel image ($I_L$ or $I_R$), each chip computes outputs ($m(c_L, \phi_L, I_L)$ or $m(c_R, \phi_R, I_R)$) corresponding to a 32 x 64 array of center positions and two phases, 0 and $-\pi/2$. All filters are designed to have the same gain, spatial frequency tuning and bandwidth. We refer to the $\phi = 0$ filter as the EVEN symmetric filter and the $\phi = -\pi/2$ filter as the ODD symmetric filter. Figure 3 shows the RF profile of the EVEN and ODD filters, which differ from a Gabor function because the function that modulates the cosine function is not a Gaussian; it decays faster at the origin and slower at the tails. This difference should not affect the resulting binocular complex cell responses significantly. Qian and Zhu [5] show that the binocular complex cell responses in the energy model are insensitive to the exact shape of the modulating envelope.

The positive and negative components of each filter output are represented by a spike rate on separate ON and OFF channels. For example, for the right eye at center position $c_R$, the EVEN-ON spike rate is proportional to $|m(c_R, 0, I_R)|^+$ and the EVEN-OFF spike rate to $|-m(c_R, 0, I_R)|^+$. Spikes are encoded on a single asynchronous digital bus using the address event representation (AER) communication protocol. The AER protocol signals the occurrence of a spike in the array by placing an address identifying the cell that spiked on the bus [10].

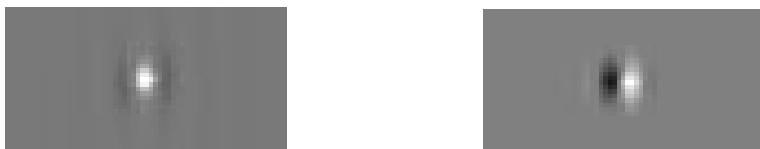

Fig. 3: The measured RF profile of the EVEN and ODD symmetric filters at the center pixel.

## 3.2 AER Address Filter

Each AER address filter extracts only those spikes corresponding to the four neurons whose RF profiles are centered at a desired retinal location and demultiplexes the spikes as voltage pulses on four separate wires. In our addressing scheme, every neuron is assigned a unique X (column) and Y (row) address. As addresses appear on the AER bus, two latches latch the row and column address of each spike, which are compared with the row and column address of the desired retinal location, which is encoded on bits 1-6 of the address. Bit 0 (the LSB) encodes the type of filter: EVEN/ODD on the row address and ON/OFF on the column address. Once the filter detects a spike from the desired retinal location, it generates a voltage pulse which is demultiplexed onto one of four output lines, depending upon the LSB of the latched row and column address.

To avoid losing events, we minimize the time the AER address filter requires to process each address by implementing it using a Xilinx XC9500 series Complex Programmable Logic Device (CPLD). We chose this series because of its speed and flexibility. The block delay in each macrocell is 7ns. The series supports in system programming, enabling rapid debugging during system design. Because the AER protocol is asynchronous, we paid particular attention to the timing in the signal path to ensure that addresses are latched correctly and to avoid glitches that could be interpreted as output spikes.

## 3.3 Binocular combination block

The binocular combination block combines eight spike trains to implement the summation operation in Eq. (1) for two phase quadrature binocular filters. To compute the two binocular filter outputs required for a zero disparity tuned cell, we first set the AER address filters so that they extract spikes from monocular neurons with the same RF centers in the left and right eyes $(\Delta c = 0)$. To compute the output of the first binocular filter B1, the binocular combination block sums the outputs of the left and right eye EVEN filters by merging spikes from the left and right EVEN-ON channels onto a positive output line, B1+ (shown in Fig. 2), and merging spikes from the left and right EVEN-OFF channels onto a negative output line, B1-. The difference between the spike rates on B1+ and B1- encodes the B1 filter output. However, the B1+ and B1- spike rates do not represent the ON (positive half-wave rectified) and OFF (negative half-wave rectified) components of the binocular filter outputs, since they may both be non-zero at the same time. To compute the output of the second filter, B2, the binocular combination block merges spikes from the left and right ODD channels similarly.

The system can also implement binocular filter outputs for neurons tuned to non-zero disparities. For position encoding, we change the relative addresses selected by the AER address filters to set $\Delta c \neq 0$, but leave the binocular combination block unchanged. If we fix the center location of the right eye RF to the center column of the chip (32), we can detect position disparities between -31 and 32 in unit pixel steps. For phase encoding, we leave the AER address filters unchanged and alter the routing in the binocular combination block. Because the RF profiles of the Gabor-type chips have two phase values, altering the routing as shown in Table 1 results in four distinct binocular filters with monocular filter phase shifts of $\Delta \phi = -\pi/2, 0, \pi/2$ and $\pi$, which correspond to the tuned far, tuned excitatory, tuned near and tuned inhibitory disparity cells identified by Poggio et al. [11]

The binocular combination block uses the same type of Xilinx CPLD as the AER filter. Inputs control the monocular phase shift of the resulting binocular filter by modifying the routing. For simplicity, we implement the merge using inclusive OR gates without arbitration. Although simultaneous spikes on the left and right channels will be merged into a single spike, the probability that this will happen is negligible, since the width of the voltage pulse that represents each spike (~32ns) is much smaller than the inter-spike intervals, which are on the order of milliseconds.

Table 1: Signal combinations for phase disparity encoding. Each table entry represents the combination of right/left eye inputs combined in a binocular output line to achieve a desired phase shift of $\Delta \phi$. We abbreviate EVEN/ODD by e/o and ON/OFF by +/-.

| | | Binocular output line | | | |
|---|---|---|---|---|---|
| | | B1+ | B1- | B2+ | B2- |
| $\Delta \phi$ | $-\pi/2$ | e+/o- | e-/o+ | o+/e+ | o-/e- |
| | $0$ | e+/e+ | e-/e- | o+/o+ | o-/o- |
| | $\pi/2$ | e+/o+ | e-/o- | o+/e- | o-/e+ |
| | $\pi$ | e+/e- | e-/e+ | o+/o- | o-/o+ |

## 3.4 Complex cell output

Since the spike rates at the four outputs of the binocular combination block are relatively low, e.g. 10-1000Hz, we implement the final steps using an 8051 microcontroller (MCU) running at 24 MHz. Integrators count the number of spikes from each channel in a fixed time window, e.g. $T = 40\text{ms}$, to estimate the average spike rate on each of the four lines. We generate the four binocular simple cell responses by positive and negative half squar-

ing the spike rate differences (B1+ - B1-) and (B2+ - B2-), and sum them to obtain the binocular complex cell output. The MCU computes one set of four simple cell and one complex cell outputs every $T$ seconds, where $T$ is the time window of the integration.

## 4 RESULTS

We use a pattern generator to supply the left and right eye inputs, which gives us precise control over the input disparity. In a loose biological analogy, we directly stimulate the optic nerve. The pattern generator simultaneously excites a pair of pixels in the left and right Gabor-type chips. The two pixels lie in the same row but are displaced by half the input disparity to the right of the center pixel in the right chip and by half the input disparity to the left of the center pixel in the left chip. The integration time window was 40ms.

Figure 4(a) shows the response of binocular complex cells tuned to three different disparities by phase encoding. The AER address filters selected spikes from the retina locations (32,16) in both chips. Consistent with theoretical predictions, the peaks of the non-zero disparity tuned cells are approximately the same height, but smaller than the peak of the zero disparity tuned filter because of the smaller size of the side peaks in the ODD filter response in comparison with the center peak in the EVEN filter. Figure 4(a) shows the response of binocular complex cells tuned to similar disparities by position encoding. The negative-disparity tuned cell combines the outputs of pixels (33,16) in the left chip and (31,16) in the right chip. The positive-disparity tuned cell combines the outputs of pixel (31,16) in the left chip and pixel (33,16) in the right chip. The zero-disparity tuned cells for position and phase encoding are identical. Theoretically, the position model should result in three identical peaks that are displaced in disparity. However, the measurements show a wide variation in the peak sizes. The responses of the phase-tuned neurons exhibit better matching, because they were all computed from the same two sets of pixel outputs. In contrast, the three position-tuned neurons combine the responses of the Gabor-type chip at six different pixels.

Decreasing the time over which we integrate the spike outputs of the binocular combinations stage results in faster disparity update. However, Figure 4(c) shows that this also increases variability in the response, when measured as a percentage of the mean response.

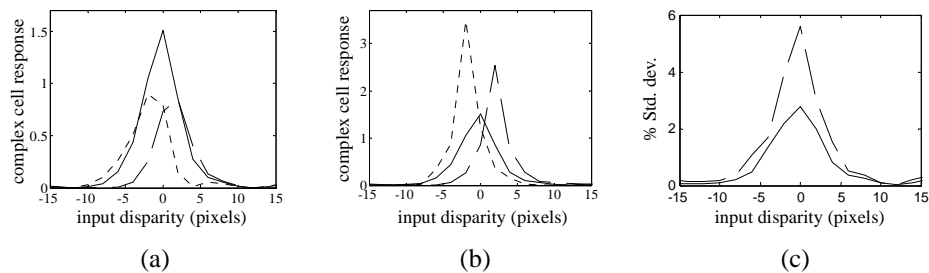

(a)                    (b)                    (c)

Fig. 4: (a) Response of three binocular complex cells tuned to three different disparities by phase encoding. (b) Response of three binocular complex cells tuned to three different disparities by position encoding. (c) Standard deviation of the response of zero disparity complex cell expressed as a percentage of the mean response at zero disparity for two integration windows of $T = 40$ms (solid line) and $T = 20$ms (dashed line). Statistics computed over 80 samples.

Although they are nominally identical, the gain, damping (bandwidth), spatial frequency and offset of neurons from different retinal locations on the same chip vary due to transis-

tor mismatch in the circuits used to implement them. We performed a numerical sensitivity analysis on the effect of variation in these parameters on the complex cell responses, by examining how much variations in them affected the locations at which the disparity tuning curves for neurons tuned to left and right disparities crossed the disparity tuning curve for the neuron tuned to zero disparity. These two locations form decision boundaries between near, zero and far disparities if we classify stimuli according to disparity tuned neuron with the maximum response. We found that the variation in the distance between these points varied much more than their centroid. Figure 5(a) shows the sensitivity coefficients for the distance between these points, where the sensitivity coefficient is defined as the percentage variation in the distance per percentage variation in a RF parameter. We consider the response to be robust to variations if the sensitivity coefficient is less than 1. In most cases, we find that the position model is less robust than the phase model.

In addition, we characterized the variability in the RF parameters for neurons from different positions on the chip. We probed the response of seven individual spiking neurons to different spatial impulse inputs and fitted parameterized Gabor-type functions to the responses. We then computed the standard deviation in the parameters across the neurons probed, which we express as a percentage of the mean value. Figure 5(b) shows that the phase model is least sensitive to variations in the parameters that vary the most.

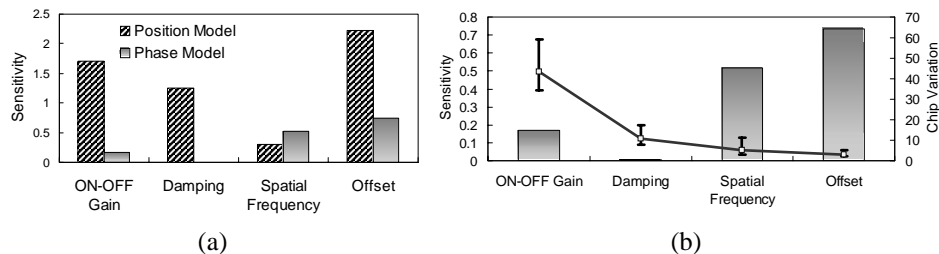

Fig. 5: (a) Sensitivity of the phase and position models to variations in the RF parameters of the neurons. (b) A comparison of the sensitivity of phase model to the variability in the RF parameters. The line indicates the percentage standard deviation in the RF parameters. Errors bars indicate the 95% confidence interval. Solid bars show the sensitivity of the phase model from (a).

## 5  CONCLUSION

We have replicated the disparity selectivity of complex cells in the visual cortex in a neuromorphic system based upon the disparity energy model. This system contains four silicon chips containing retinotopic arrays of neurons which communicate via the AER communication protocol, as well as circuits that combine the outputs of these chips to generate the response of a model binocular complex cell. We exploit the capability of AER protocol for point to point communication, as well as the ability to reroute spikes.

Our measurements indicate that our binocular complex cells are disparity selective and that their selectivity can be adjusted through both position and phase encoding. However, the relative responses of neurons tuned by phase encoding exhibit better matching than the relative responses of neurons tuned by position encoding, because neurons tuned to different disparities by position encoding integrate outputs from different pixels while neurons tuned by phase encoding integrate output from the same pixels.

This implementation is an initial step towards the development of a multi-chip neuromorphic system capable of extracting depth information about the visual environment using silicon neurons with physiologically-based functionality. The next step will be to extend

the system from a single disparity tuned neuron to a set of retinotopic arrays of disparity tuned neurons. In order to do this, we will develop a mixed analog-digital chip whose architecture will be similar to that of the orientation tuned chip, which will combine the outputs from left and right eye orientation-tuned chips to compute an array of neurons tuned to the same disparity but different retinal locations. The tuned disparity can be controlled by address remapping, so additional copies of the same chip could represent neurons tuned to other disparities. This chip will increase the number of neurons we compute simultaneously, as well as decreasing the power consumption required to compute each neuron. In the current implementation, the digital circuits required to combine the monocular responses consume 1.2W. In contrast, the Gabor chips and their associated external bias and interface circuits consume only 62mW, with only about 4mW required for each Gabor chip. We expect the power consumption of the binocular combination chip to be comparable. Computing the neuron outputs in parallel will enable us to investigate the roles of additional processing steps such as pooling [5], [6] and normalization [12], [13].

## Acknowledgements

This work was supported in part by the Hong Kong Research Grants Council under Grant HKUST6218/01E. It was inspired by a project with Y. Miyawaki at the 2002 Telluride Neuromorphic Workshop. The authors would like to thank K. A. Boahen for helpful discussions and for supplying the receiver board used in this work, and T. Choi for his assistance in building the system.

## References

[1]    Barlow, H. B., Blackemore, C., & Pettigrew, J. D. (1967) The neural mechanism of binocular depth discrimination. *J. Physiol. Lond.*, **193**, 327-342.

[2]    Ohzawa, I., Deangelis, G. C., & Freeman, R. D. (1990) Stereoscopic depth discrimination in the visual cortex: neurons ideally suited as disparity detectors. *Science*, **249**, 1037-1041.

[3]    Cummings, B. G. & Parker, A. J. (1997) Responses of primary visual cortical neurons to binocular disparity without depth perception. *Nature*, **389**, 280-283.

[4]    Anzai, A., Ohzawa, I., and Freeman, R. D. (1999a) Neural mechanisms for encoding binocular disparity: position vs. phase. *J. Neurophysiol.,* **82,** 874-890.

[5]    Qian, N., & Zhu, Y. (1997) Physiological computation of binocular disparity. *Vision Res.*, **37**, 1811-1827.

[6]    Fleet, D. J., Wagner, H., & Heeger, D. J. (1996) Neural encoding of binocular disparity: energy models, position shifts and phase shifts. *Vision Res.*, **36**, 1839-57.

[7]    Anzai, A., Ohzawa, I., and Freeman, R. D. (1999b) Neural mechanisms for processing binocular information I. Simple cells. *J. Neurophysiol.,* **82**, 891-908.

[8]    Anzai, A., Ohzawa, I., and Freeman, R. D. (1999c) Neural mechanisms for processing binocular information II. Complex cells. *J. Neurophysiol.*, **82**, 909-924.

[9]    Choi, T. Y. W., Shi, B. E., & Boahen, K. (2003) An Orientation Selective 2D AER Transceiver, *Proceedings of the IEEE Intl. Conf. on Circuits and Systems*, **4**, 800-803.

[10]   Boahen, K. A. (2000) Point-to-point connectivity between neuromorphic chips using address events. *IEEE Transactions on Circuits and Systems-II: Analog and Digital Signal Processing*, **47**, 416-434.

[11]   Poggio, G. F., Motter, B. C. Squatrito, S., & Trotter, Y. (1985) Responses of neurons in visual cortex (V1 and V2) of the alert macaque to dynamic random-dot stereograms. *Vision Research*, **25**, 397-406.

[12]   Albrecht, D. G. & Geisler, W. S. (1991) Motion selectivity and the contrast response functions of simple cells in the visual cortex, *Visual Neuroscience*, **7**, 531-546.

[13]   Heeger, D. J. (1992). Normalization of cell responses in cat striate cortex. *Visual Neuroscience*, **9**, 181-197.
